# Learning Horizontal Connections in a Sparse Coding Model of Natural Images

**Pierre J. Garrigues**
Department of EECS
Redwood Center for Theoretical Neuroscience
Univ. of California, Berkeley
Berkeley, CA 94720
garrigue@eecs.berkeley.edu

**Bruno A. Olshausen**
Helen Wills Neuroscience Inst.
School of Optometry
Redwood Center for Theoretical Neuroscience
Univ. of California, Berkeley
Berkeley, CA 94720
baolshausen@berkeley.edu

## Abstract

It has been shown that adapting a dictionary of basis functions to the statistics of natural images so as to maximize sparsity in the coefficients results in a set of dictionary elements whose spatial properties resemble those of V1 (primary visual cortex) receptive fields. However, the resulting sparse coefficients still exhibit pronounced statistical dependencies, thus violating the independence assumption of the sparse coding model. Here, we propose a model that attempts to capture the dependencies among the basis function coefficients by including a pairwise coupling term in the prior over the coefficient activity states. When adapted to the statistics of natural images, the coupling terms learn a combination of facilitatory and inhibitory interactions among neighboring basis functions. These learned interactions may offer an explanation for the function of horizontal connections in V1 in terms of a prior over natural images.

## 1 Introduction

Over the last decade, mathematical explorations into the statistics of natural scenes have led to the observation that these scenes, as complex and varied as they appear, have an underlying structure that is sparse [1]. That is, one can learn a possibly overcomplete basis set such that only a small fraction of the basis functions is necessary to describe a given image, where the operation to infer this sparse representation is non-linear. This approach is known as sparse coding. Exploiting this structure has led to advances in our understanding of how information is represented in the visual cortex, since the learned basis set is a collection of oriented, Gabor-like filters that resemble the receptive fields in primary visual cortex (V1). The approach of using sparse coding to infer sparse representations of unlabeled data is useful for classification as shown in the framework of self-taught learning [2]. Note that classification performance relies on finding "hard-sparse" representations where a few coefficients are nonzero while all the others are exactly zero.

An assumption of the sparse coding model is that the coefficients of the representation are independent. However, in the case of natural images, this is not the case. For example, the coefficients corresponding to quadrature pair or colinear Gabor filters are not independent. This has been shown and modeled in the early work of [3], in the case of the responses of model complex cells [4], feedforward responses of wavelet coefficients [5, 6, 7] or basis functions learned using independent component analysis [8, 9]. These dependencies are informative and exploiting them leads to improvements in denoising performance [5, 7].

We develop here a generative model of image patches that does not make the independence assumption. The prior over the coefficients is a mixture of a Gaussian when the corresponding basis

function is active, and a delta function centered at zero when it is silent as in [10]. We model the binary variables or "spins" that control the activation of the basis functions with an Ising model, whose coupling weights model the dependencies among the coefficients. The representations inferred by this model are also "hard-sparse", which is a desirable feature [2].

Our model is motivated in part by the architecture of the visual cortex, namely the extensive network of horizontal connections among neurons in V1 [11]. It has been hypothesized that they facilitate contour integration [12] and are involved in computing border ownership [13]. In both of these models the connections are set *a priori* based on geometrical properties of the receptive fields. We propose here to learn the connection weights in an unsupervised fashion. We hope with our model to gain insight into the the computations performed by this extensive collateral system and compare our findings to known physiological properties of these horizontal connections. Furthermore, a recent trend in neuroscience is to model networks of neurons using Ising models, and it has been shown to predict remarkably well the statistics of groups of neurons in the retina [14]. Our model gives a prediction for what is expected if one fits an Ising model to future multi-unit recordings in V1.

## 2   A non-factorial sparse coding model

Let $x \in \mathbb{R}^n$ be an image patch, where the $x_i$'s are the pixel values. We propose the following generative model:

$$x = \Phi a + \nu = \sum_{i=1}^{m} a_i \varphi_i + \nu,$$

where $\Phi = [\varphi_1 \ldots \varphi_m] \in \mathbb{R}^{n \times m}$ is an overcomplete transform or basis set, and the columns $\varphi_i$ are its basis functions. $\nu \sim \mathcal{N}(0, \epsilon^2 I_n)$ is small Gaussian noise. Each coefficient $a_i = \frac{s_i+1}{2} u_i$ is a Gaussian scale mixture (GSM). We model the multiplier $s$ with an Ising model, i.e. $s \in \{-1, 1\}^m$ has a Boltzmann-Gibbs distribution $p(s) = \frac{1}{Z} e^{\frac{1}{2} s^T W s + b^T s}$, where $Z$ is the normalization constant. If the spin $s_i$ is down ($s_i = -1$), then $a_i = 0$ and the basis function $\varphi_i$ is silent. If the spin $s_i$ is up ($s_i = 1$), then the basis function is active and the analog value of the coefficient $a_i$ is drawn from a Gaussian distribution with $u_i \sim \mathcal{N}(0, \sigma_i^2)$. The prior on $a$ can thus be described as a "hard-sparse" prior as it is a mixture of a point mass at zero and a Gaussian.

The corresponding graphical model is shown in Figure 1. It is a chain graph since it contains both undirected and directed edges. It bears similarities to [15], which however does not have the intermediate layer $a$ and is not a sparse coding model. To sample from this generative model, one first obtains a sample $s$ from the Ising model, then samples coefficients $a$ according to $p(a \mid s)$, and then $x$ according to $p(x \mid a) \sim \mathcal{N}(\Phi a, \epsilon^2 I_n)$.

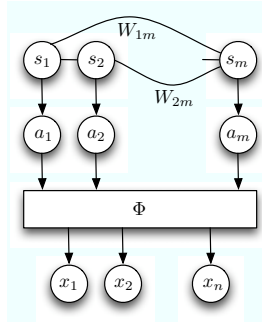

Figure 1: Proposed graphical model

The parameters of the model to be learned from data are $\theta = (\Phi, (\sigma_i^2)_{i=1..m}, W, b)$. This model does not make any assumption about which linear code $\Phi$ should be used, and about which units should exhibit dependencies. The matrix $W$ of the interaction weights in the Ising model describes these dependencies. $W_{ij} > 0$ favors positive correlations and thus corresponds to an excitatory connection, whereas $W_{ij} < 0$ corresponds to an inhibitory connection. A local magnetic field $b_i < 0$ favors the spin $s_i$ to be down, which in turn makes the basis function $\varphi_i$ mostly silent.

## 3 Inference and learning

### 3.1 Coefficient estimation

We describe here how to infer the representation $a$ of an image patch $x$ in our model. To do so, we first compute the maximum a posteriori (MAP) multiplier $s$ (see Section 3.2). Indeed, a GSM model reduces to a linear-Gaussian model conditioned on the multiplier $s$, and therefore the estimation of $a$ is easy once $s$ is known.

Given $s = \hat{s}$, let $\Gamma = \{i : \hat{s}_i = 1\}$ be the set of active basis functions. We know that $\forall i \notin \Gamma$, $a_i = 0$. Hence, we have $x = \Phi_\Gamma a_\Gamma + \nu$, where $a_\Gamma = (a_i)_{i \in \Gamma}$ and $\Phi_\Gamma = [(\varphi_i)_{i \in \Gamma}]$. The model reduces thus to linear-Gaussian, where $a_\Gamma \sim \mathcal{N}(0, H = \text{diag}((\sigma_i^2)_{i \in \Gamma}))$. We have $a_\Gamma \mid x, \hat{s} \sim \mathcal{N}(\mu, K)$, where $K = (\epsilon^{-2} \Phi_\Gamma \Phi_\Gamma^T + H^{-1})^{-1}$ and $\mu = \epsilon^{-2} K \Phi_\Gamma^T x$. Hence, conditioned on $x$ and $\hat{s}$, the Bayes Least-Square (BLS) and maximum a posteriori (MAP) estimators of $a_\Gamma$ are the same and given by $\mu$.

### 3.2 Multiplier estimation

The MAP estimate of $s$ given $x$ is given by $\hat{s} = \arg\max_s p(s \mid x)$. Given $s$, $x$ has a Gaussian distribution $\mathcal{N}(0, \Sigma)$, where $\Sigma = \epsilon^2 I_n + \sum_{i \,:\, s_i=1} \sigma_i^2 \varphi_i \varphi_i^T$. Using Bayes' rule, we can write $p(s \mid x) \propto p(x \mid s)p(s) \propto e^{-E_x(s)}$, where

$$E_x(s) = \frac{1}{2}x^T \Sigma^{-1} x + \frac{1}{2}\log\det\Sigma - \frac{1}{2}s^T W s - b^T s.$$

We can thus compute the MAP estimate using Gibbs sampling and simulated annealing. In the Gibbs sampling procedure, the probability that node $i$ changes its value from $s_i$ to $\bar{s}_i$ given $x$, all the other nodes $s_{\neg i}$ and at temperature $T$ is given by

$$p(s_i \to \bar{s}_i | s_{\neg i}, x) = \left(1 + \exp\left(-\frac{\Delta E_x}{T}\right)\right)^{-1},$$

where $\Delta E_x = E_x(s_i, s_{\neg i}) - E_x(\bar{s}_i, s_{\neg i})$. Note that computing $E_x$ requires the inverse and the determinant of $\Sigma$, which is expensive. Let $\bar{\Sigma}$ and $\Sigma$ be the covariance matrices corresponding to the proposed state $(\bar{s}_i, s_{\neg i})$ and current state $(s_i, s_{\neg i})$ respectively. They differ only by a rank 1 matrix, i.e. $\bar{\Sigma} = \Sigma + \alpha \varphi_i \varphi_i^T$, where $\alpha = \frac{1}{2}(\bar{s}_i - s_i)\sigma_i^2$. Therefore, to compute $\Delta E_x$ we can take advantage of the Sherman-Morrison formula

$$\bar{\Sigma}^{-1} = \Sigma^{-1} - \alpha \Sigma^{-1} \varphi_i (1 + \alpha \varphi_i^T \Sigma^{-1} \varphi_i)^{-1} \varphi_i^T \Sigma^{-1} \tag{1}$$

and of a similar formula for the $\log\det$ term

$$\log\det\bar{\Sigma} = \log\det\Sigma + \log\left(1 + \alpha \varphi_i^T \Sigma^{-1} \varphi_i\right). \tag{2}$$

Using (1) and (2) $\Delta E_x$ can be written as

$$\Delta E_x = \frac{1}{2}\frac{\alpha(x^T \Sigma^{-1} \varphi_i)^2}{1 + \alpha \varphi_i^T \Sigma^{-1} \varphi_i} - \frac{1}{2}\log\left(1 + \alpha \varphi_i^T \Sigma^{-1} \varphi_i\right) + (\bar{s}_i - s_i)\left(\sum_{j \neq i} W_{ij} s_j + b_i\right).$$

The transition probabilities can thus be computed efficiently, and if a new state is accepted we update $\Sigma$ and $\Sigma^{-1}$ using (1).

### 3.3 Model estimation

Given a dataset $\mathcal{D} = \{x^{(1)}, \ldots, x^{(N)}\}$ of image patches, we want to learn the parameters $\theta = (\Phi, (\sigma_i^2)_{i=1..m}, W, b)$ that offer the best explanation of the data. Let $p^*(x) = \frac{1}{N}\sum_{i=1}^N \delta(x - x^{(i)})$ be the empirical distribution. Since in our model the variables $a$ and $s$ are latent, we use a variational expectation maximization algorithm [16] to optimize $\theta$, which amounts to maximizing a lower bound on the log-likelihood derived using Jensen's inequality

$$\log p(x \mid \theta) \;\geq\; \sum_s \int_a q(a, s \mid x) \log \frac{p(x, a, s \mid \theta)}{q(a, s \mid x)} da,$$

where $q(a, s \mid x)$ is a probability distribution. We restrict ourselves to the family of point mass distributions $\mathcal{Q} = \{q(a, s \mid x) = \delta(a - \hat{a})\delta(s - \hat{s})\}$, and with this choice the lower bound on the log-likelihood of $\mathcal{D}$ can be written as

$$
\begin{aligned}
\mathcal{L}(\theta, q) &= \mathbb{E}_{p^*}\left[\log p(x, \hat{a}, \hat{s} \mid \theta)\right] \quad\quad\quad\quad\quad\quad\quad\quad\quad\quad\quad\quad (3) \\
&= \underbrace{\mathbb{E}_{p^*}\left[\log p(x \mid \hat{a}, \Phi)\right]}_{\mathcal{L}_\Phi} + \underbrace{\mathbb{E}_{p^*}\left[\log p(\hat{a} \mid \hat{s}, (\sigma_i^2)_{i=1..m})\right]}_{\mathcal{L}_\sigma} + \underbrace{\mathbb{E}_{p^*}\left[\log p(\hat{s} \mid W, b)\right]}_{\mathcal{L}_{W,b}}.
\end{aligned}
$$

We perform coordinate ascent in the objective function $\mathcal{L}(\theta, q)$.

### 3.3.1 Maximization with respect to $q$

We want to solve $\max_{q \in \mathcal{Q}} \mathcal{L}(\theta, q)$, which amounts to finding $\arg\max_{a,s} \log p(x, a, s)$ for every $x \in \mathcal{D}$. This is computationally expensive since $s$ is discrete. Hence, we introduce two phases in the algorithm.

In the first phase, we infer the coefficients in the usual sparse coding model where the prior over $a$ is factorial, i.e. $p(a) = \prod_i p(a_i) \propto \prod_i \exp\{-\lambda S(a_i)\}$. In this setting, we have

$$
\hat{a} = \arg\max_a p(x|a) \prod_i e^{-\lambda S(a_i)} = \arg\min_a \frac{1}{2\epsilon^2} \|x - \Phi a\|_2^2 + \lambda \sum_i S(a_i). \quad\quad (4)
$$

With $S(a_i) = |a_i|$, (4) is known as basis pursuit denoising (BPDN) whose solution has been shown to be such that many coefficient of $\hat{a}$ are exactly zero [17]. This allows us to recover the sparsity pattern $\hat{s}$, where $\hat{s}_i = 2.\mathbf{1}[\hat{a}_i \neq 0] - 1 \; \forall i$. BPDN can be solved efficiently using a competitive algorithm [18]. Another possible choice is $S(a_i) = \mathbf{1}[a_i \neq 0]$ ($p(a_i)$ is not a proper prior though), where (4) is combinatorial and can be solved approximately using orthogonal matching pursuits (OMP) [19].

After several iterations of coordinate ascent and convergence of $\theta$ using the above approximation, we enter the second phase of the algorithm and refine $\theta$ by using the GSM inference described in Section 3.1 where $\hat{s} = \arg\max p(s|x)$ and $\hat{a} = \mathbb{E}[a \mid \hat{s}, x]$.

### 3.3.2 Maximization with respect to $\theta$

We want to solve $\max_\theta \mathcal{L}(\theta, q)$. Our choice of variational posterior allowed us to write the objective function as the sum of the three terms $\mathcal{L}_\Phi$, $\mathcal{L}_\sigma$ and $\mathcal{L}_{W,b}$ (3), and hence to decouple the variables $\Phi$, $(\sigma_i^2)_{i=1..m}$ and $(W, b)$ of our optimization problem.

**Maximization of $\mathcal{L}_\Phi$.** Note that $\mathcal{L}_\Phi$ is the same objective function as in the standard sparse coding problem when the coefficients $a$ are fixed. Let $\{\hat{a}^{(i)}, \hat{s}^{(i)}\}$ be the coefficients and multipliers corresponding to $x^{(i)}$. We have

$$
\mathcal{L}_\Phi = -\frac{1}{2\epsilon^2} \sum_{i=1}^N \|x^{(i)} - \Phi\hat{a}^{(i)}\|_2^2 - \frac{Nn}{2} \log 2\pi\epsilon^2.
$$

We add the constraint that $\|\varphi_i\|_2 \leq 1$ to avoid the spurious solution where the norm of the basis functions grows and the coefficients tend to 0. We solve this $\ell_2$ constrained least-square problem using the Lagrange dual as in [20].

**Maximization of $\mathcal{L}_\sigma$.** The problem of estimating $\sigma_i^2$ is a standard variance estimation problem for a 0-mean Gaussian random variable, where we only consider the samples $\hat{a}_i$ such that the spin $\hat{s}_i$ is equal to 1, i.e.

$$
\sigma_i^2 = \frac{1}{\text{card}\{k \; : \; \hat{s}_i^{(k)} = 1\}} \sum_{k \; : \; \hat{s}_i^{(k)}=1} (\hat{a}_i^{(k)})^2.
$$

**Maximization of $\mathcal{L}_{W,b}$.** This problem is tantamount to estimating the parameters of a fully visible Boltzmann machine [21] which is a convex optimization problem. We do gradient ascent in $\mathcal{L}_{W,b}$, where the gradients are given by $\frac{\partial \mathcal{L}_{W,b}}{\partial W_{ij}} = -\mathbb{E}_{p^*}[s_i s_j] + \mathbb{E}_p[s_i s_j]$ and $\frac{\partial \mathcal{L}_{W,b}}{\partial b_i} = -\mathbb{E}_{p^*}[s_i] + \mathbb{E}_p[s_i]$. We use Gibbs sampling to obtain estimates of $\mathbb{E}_p[s_i s_j]$ and $\mathbb{E}_p[s_i]$.

Note that since computing the parameters $(\hat{a}, \hat{s})$ of the variational posterior in phase 1 only depends on $\Phi$, we first perform several steps of coordinate ascent in $(\Phi, q)$ until $\Phi$ has converged, which is the same as in the usual sparse coding algorithm. We then maximize $\mathcal{L}_\sigma$ and $L_{W,b}$, and after that we enter the second phase of the algorithm.

## 4    Recovery of the model parameters

Although the learning algorithm relies on a method where the family of variational posteriors $q(a, s \mid x)$ is quite limited, we argue here that if data $\mathcal{D} = \{x^{(1)}, \ldots, x^{(N)}\}$ is being sampled according to parameters $\theta_0$ that obey certain conditions that we describe now, then our proposed learning algorithm is able to recover $\theta_0$ with good accuracy using phase 1 only.

Let $\eta$ be the coherence parameter of the basis set which equals the maximum absolute inner product between two distinct basis functions. It has been shown that given a signal that is a sparse linear combination of $p$ basis functions, BP and OMP will identify the optimal basis functions and their coefficients provided that $p < \frac{1}{2}(\eta^{-1} + 1)$, and the sparsest representation of the signal is unique [19]. Similar results can be derived when noise is present ($\epsilon > 0$) [22], but we restrict ourselves to the noiseless case for simplicity. Let $\|s\|_\uparrow$ be the number of spins that are up. We require $(W_0, b_0)$ to be such that $Pr\left(\|s\|_\uparrow < \frac{1}{2}(\eta^{-1} + 1)\right) \approx 1$, which can be enforced by imposing strong negative biases. A data point $x^{(i)} \in \mathcal{D}$ thus has a high probability of yielding a unique sparse representation in the basis set $\Phi$. Provided that we have a good estimate of $\Phi$ we can recover its sparse representation using OMP or BP, and therefore identify $s^{(i)}$ that was used to originally sample $x^{(i)}$. That is we recover with high probability all the samples from the Ising model used to generate $\mathcal{D}$, which allows us to recover $(W_0, b_0)$.

We provide for illustration a simple example of model recovery where $n = 7$ and $m = 8$. Let $(e_1, \ldots, e_7)$ be an orthonormal basis in $\mathbb{R}^7$. We let $\Phi_0 = [e_1, \ldots e_7, \frac{1}{\sqrt{7}} \sum_i e_i]$. We fix the biases $b_0$ at $-1.2$ such that the model is sufficiently sparse as shown by the histogram of $\|s\|_\uparrow$ in Figure 2, and the weights $W_0$ are sampled according to a Gaussian distribution. The variance parameters $\sigma_0$ are fixed to 1. We then generate synthetic data by sampling 100000 data from this model using $\theta_0$. We then estimate $\theta$ from this synthetic data using the variational method described in Section 3 using OMP and phase 1 only. We found that the basis functions are recovered exactly (not shown), and that the parameters of the Ising model are recovered with high accuracy as shown in Figure 2.

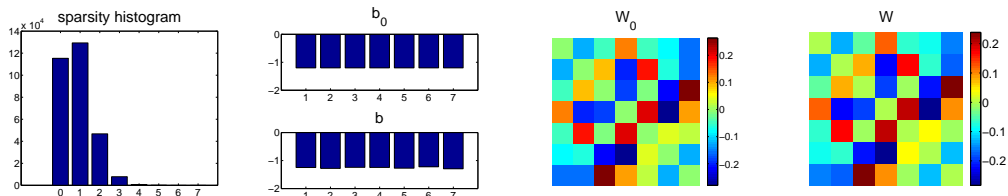

Figure 2: Recovery of the model. The histogram of $\|s\|_\uparrow$ is such that the model is sparse. The parameters $(W, b)$ learned from synthetic data are close to the parameters $(W_0, b_0)$ from which this data was generated.

## 5    Results for natural images

We build our training set by randomly selecting $16 \times 16$ image patches from a standard set of 10 $512 \times 512$ whitened images as in [1]. It has been shown that change of luminance or contrast have little influence on the structure of natural scenes [23]. As our goal is to uncover this structure, we subtract from each patch its own mean and divide it by its standard deviation such that our dataset is contrast normalized (we do not consider the patches whose variance is below a small threshold). We fix the number of basis functions to 256. In the second phase of the algorithm we only update $\Phi$, and we have found that the basis functions do not change dramatically after the first phase.

Figure 3 shows the learned parameters $\Phi$, $\sigma$ and $b$. The basis functions resemble Gabor filters at a variety of orientations, positions and scales. We show the weights $W$ in Figure 4 according to

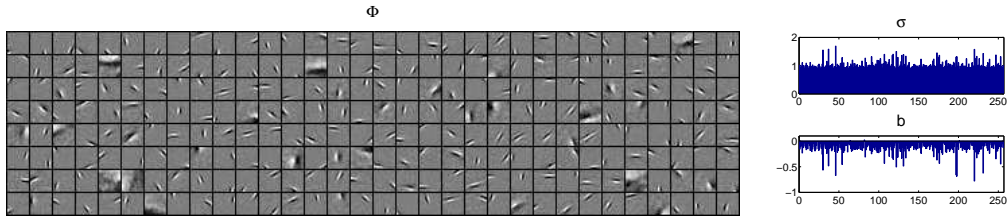

Figure 3: On the left is shown the entire set of basis functions $\Phi$ learned on natural images. On the right are the learned variances $(\sigma_i^2)_{i=1..m}$ (top) and the biases $b$ in the Ising model (bottom).

the spatial properties (position, orientation, length) of the basis functions that are linked together by them. Each basis function is denoted by a bar that indicates its position, orientation, and length within the $16 \times 16$ patch.

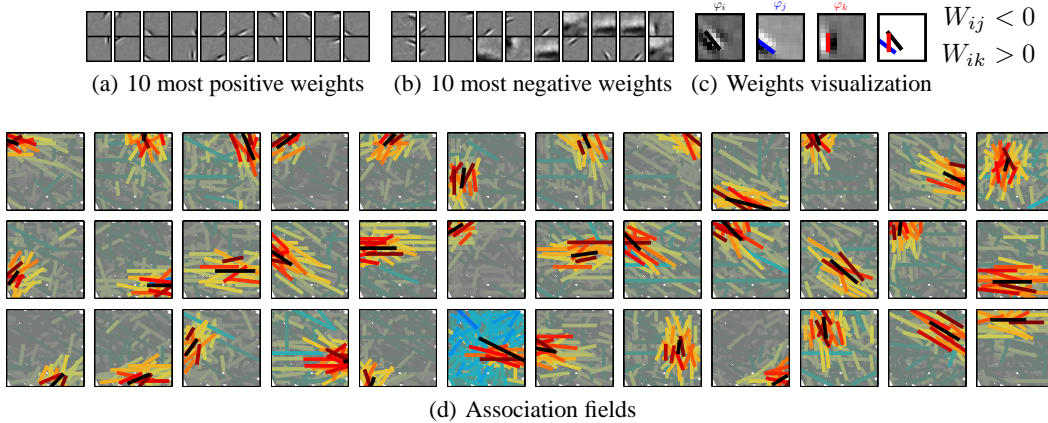

(a) 10 most positive weights    (b) 10 most negative weights    (c) Weights visualization

(d) Association fields

Figure 4: (a) (resp. (b)) shows the basis function pairs that share the strongest positive (resp. negative) weights ordered from left to right. Each subplot in (d) shows the association field for a basis function $\varphi_i$ whose position and orientation are denoted by the black bar. The horizontal connections $(W_{ij})_{j \neq i}$ are displayed by a set of colored bars whose orientation and position denote those of the basis functions $\varphi_j$ to which they correspond, and the color denotes the connection strength (see (c)). We show a random selection of 36 association fields, see *www.eecs.berkeley.edu/ garrigue/nips07.html* for the whole set.

We observe that the connections are mainly local and connect basis functions at a variety of orientations. The histogram of the weights (see Figure 5) shows a long positive tail corresponding to a bias toward facilitatory connections. We can see in Figure 4a,b that the 10 most "positive" pairs have similar orientations, whereas the majority of the 10 most "negative" pairs have dissimilar orientations. We compute for a basis function the average number of basis functions sharing with it a weight larger than 0.01 as a function of their orientation difference in four bins, which we refer to as the "orientation profile" in Figure 5. The error bars are a standard deviation. The resulting orientation profile is consistent with what has been observed in physiological experiments [24, 25].

We also show in Figure 5 the tradeoff between the signal to noise ratio (SNR) of an image patch $x$ and its reconstruction $\Phi\hat{a}$, and the $\ell_0$ norm of the representation $\|\hat{a}\|_0$. We consider $\hat{a}$ inferred using both the Laplacian prior and our proposed prior. We vary $\lambda$ (see Equation (4)) and $\epsilon$ respectively, and average over 1000 patches to obtain the two tradeoff curves. We see that at similar SNR the representations inferred by our model are more sparse by about a factor of 2, which bodes well for compression. We have also compared our prior for tasks such as denoising and filling-in, and have found its performance to be similar to the factorial Laplacian prior even though it does not exploit the dependencies of the code. One possible explanation is that the greater sparsity of our inferred representations makes them less robust to noise. Thus we are currently investigating whether this

property may instead have advantages in the self-taught learning setting in improving classification performance.

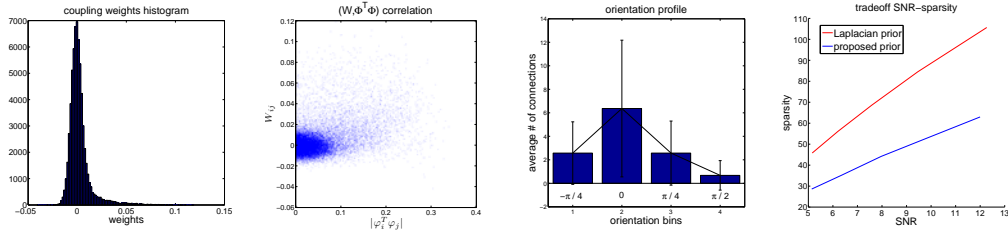

Figure 5: Properties of the weight matrix $W$ and comparison of the tradeoff curve SNR - $\ell_0$ norm between a Laplacian prior over the coefficients and our proposed prior.

To access how much information is captured by the second-order statistics, we isolate a group $(\varphi_i)_{i \in \Lambda}$ of 10 basis functions sharing strong weights. Given a collection of image patches that we sparsify using (4), we obtain a number of spins $(\hat{s}_i)_{i \in \Lambda}$ from which we can estimate the empirical distribution $p_{emp}$, the Boltzmann-Gibbs distribution $p_{Ising}$ consistent with first and second order correlations, and the factorial distribution $p_{fact}$ (i.e. no horizontal connections) consistent with first order correlations. We can see in Figure 6 that the Ising model produces better estimates of the empirical distribution, and results in better coding efficiency since $KL(p_{emp}||p_{Ising}) = .02$ whereas $KL(p_{emp}||p_{fact}) = .1$.

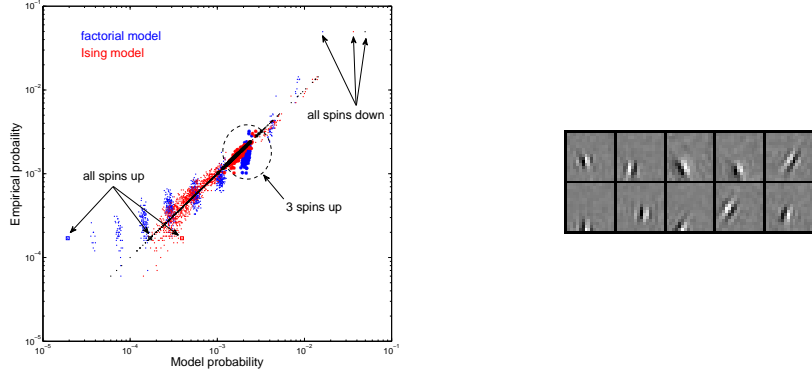

Figure 6: Model validation for a group of 10 basis functions (right). The empirical probabilities of the $2^{10}$ patterns of activation are plotted against the probabilities predicted by the Ising model (red), the factorial model (blue), and their own values (black). These patterns having exactly three spins up are circled. The prediction of the Ising model is noticably better than that of the factorial model.

## 6 Discussion

In this paper, we proposed a new sparse coding model where we include pairwise coupling terms among the coefficients to capture their dependencies. We derived a new learning algorithm to adapt the parameters of the model given a data set of natural images, and we were able to discover the dependencies among the basis functions coefficients. We showed that the learned connection weights are consistent with physiological data. Furthermore, the representations inferred in our model have greater sparsity than when they are inferred using the Laplacian prior as in the standard sparse coding model. Note however that we have not found evidence that these horizontal connections facilitate contour integration, as they do not primarily connect colinear basis functions. Previous models in the literature simply assume these weights according to prior intuitions about the function of horizontal connections [12, 13]. It is of great interest to develop new models and unsupervised learning schemes possibly involving attention that will help us understand the computational principles underlying contour integration in the visual cortex.

# References

[1] B.A. Olshausen and D. J. Field. Emergence of simple-cell receptive field properties by learning a sparse code for natural images. *Nature*, 381(6583):607–609, June 1996.

[2] R. Raina, A. Battle, H. Lee, B. Packer, and A.Y. Ng. Self-taught learning: Transfer learning from unlabeled data. *Proceedings of the Twenty-fourth International Conference on Machine Learning*, 2007.

[3] G. Zetzsche and B. Wegmann. The atoms of vision: Cartesian or polar? *J. Opt. Soc. Am.*, 16(7):1554–1565, 1999.

[4] P. Hoyer and A. Hyvärinen. A multi-layer sparse coding network learns contour coding from natural images. *Vision Research*, 42:1593–1605, 2002.

[5] M.J. Wainwright, E.P. Simoncelli, and A.S. Willsky. Random cascades on wavelet trees and their use in modeling and analyzing natural imagery. *Applied and Computational Harmonic Analysis*, 11(1):89–123, July 2001.

[6] O. Schwartz, T. J. Sejnowski, and P. Dayan. Soft mixer assignment in a hierarchical generative model of natural scene statistics. *Neural Comput*, 18(11):2680–2718, November 2006.

[7] S. Lyu and E. P. Simoncelli. Statistical modeling of images with fields of gaussian scale mixtures. In *Advances in Neural Computation Systems (NIPS)*, Vancouver, Canada, 2006.

[8] A. Hyvärinen, P.O. Hoyer, J. Hurri, and M. Gutmann. Statistical models of images and early vision. *Proceedings of the Int. Symposium on Adaptive Knowledge Representation and Reasoning (AKRR2005)*, Espoo, Finland, 2005.

[9] Y. Karklin and M.S. Lewicki. A hierarchical bayesian model for learning non-linear statistical regularities in non-stationary natural signals. *Neural Computation*, 17(2):397–423, 2005.

[10] B.A. Olshausen and K.J. Millman. Learning sparse codes with a mixture-of-gaussians prior. *Advances in Neural Information Processing Systems, 12*, 2000.

[11] D. Fitzpatrick. The functional organization of local circuits in visual cortex: insights from the study of tree shrew striate cortex. *Cerebral Cortex*, 6:329–41, 1996.

[12] O. Ben-Shahar and S. Zucker. Geometrical computations explain projection patterns of long-range horizontal connections in visual cortex. *Neural Comput*, 16(3):445–476, March 2004.

[13] L. Zhaoping. Border ownership from intracortical interactions in visual area v2. *Neuron*, 47:143–153, 2005.

[14] E. Schneidman, M.J. Berry, R. Segev, and W. Bialek. Weak pairwise correlations imply strongly correlated network states in a neural population. *Nature*, April 2006.

[15] G. Hinton, S. Osindero, and K. Bao. Learning causally linked markov random fields. *Artificial Intelligence and Statistics*, Barbados, 2005.

[16] M.I. Jordan, Z. Ghahramani, T. Jaakkola, and L.K. Saul. An introduction to variational methods for graphical models. *Learning in Graphical Models*, Cambridge, MA: MIT Press, 1999.

[17] S.S. Chen, D.L. Donoho, and M.A. Saunders. Atomic decomposition by basis pursuit. *SIAM Review*, 43(1):129–159, 2001.

[18] C.J. Rozell, D.H. Johnson, R.G. Baraniuk, and B.A. Olshausen. Neurally plausible sparse coding via competitive algorithms. In *Proceedings of the Computational and Systems Neuroscience (Cosyne) meeting*, Salt Lake City, UT, February 2007.

[19] J.A. Tropp. Greed is good: algorithmic results for sparse approximation. *IEEE Transactions on Information Theory*, 50(10):2231–2242, 2004.

[20] H. Lee, A. Battle, R. Raina, and A.Y. Ng. Efficient sparse coding algorithms. In *Advances in Neural Information Processing Systems 19*, pages 801–808. MIT Press, Cambridge, MA, 2007.

[21] D.H. Ackley, G.E. Hinton, and T.J. Sejnowski. A learning algorithm for boltzmann machines. *Cognitive Science*, 9(1):147–169, 1985.

[22] J.A. Tropp. Just relax: convex programming methods for identifying sparse signals in noise. *IEEE Transactions on Information Theory*, 52(3):1030–1051, 2006.

[23] Z. Wang, A.C. Bovik, and E.P. Simoncelli. Structural approaches to image quality assessment. In Alan Bovik, editor, *Handbook of Image and Video Processing*, chapter 8.3, pages 961–974. Academic Press, May 2005. 2nd edition.

[24] R. Malach, Y. Amir, M. Harel, and A. Grinvald. Relationship between intrinsic connections and functional architecture revealed by optical imaging and in vivo targeted biocytin injections in primate striate cortex. *Proc. Natl. Acad. Sci. U.S.A.*, 82:935–939, 1993.

[25] W. Bosking, Y. Zhang, B. Schofield, and D. Fitzpatrick. Orientation selectivity and the arrangement of horizontal connections in the tree shrew striate cortex. *J. Neuroscience*, 17(6):2112–2127, 1997.

